# The Correlated Correspondence Algorithm for Unsupervised Registration of Nonrigid Surfaces

**Dragomir Anguelov**[1]**, Praveen Srinivasan**[1]**, Hoi-Cheung Pang**[1]**,**
**Daphne Koller**[1]**, Sebastian Thrun**[1]**, James Davis**[2] *
[1] Stanford University, Stanford, CA 94305
[2] University of California, Santa Cruz, CA 95064
e-mail:{drago,praveens,hcpang,koller,thrun,jedavis}@cs.stanford.edu

## Abstract

We present an unsupervised algorithm for registering 3D surface scans of an object undergoing significant deformations. Our algorithm does not need markers, nor does it assume prior knowledge about object shape, the dynamics of its deformation, or scan alignment. The algorithm registers two meshes by optimizing a joint probabilistic model over all point-to-point correspondences between them. This model enforces preservation of local mesh geometry, as well as more global constraints that capture the preservation of geodesic distance between corresponding point pairs. The algorithm applies even when one of the meshes is an incomplete range scan; thus, it can be used to automatically fill in the remaining surfaces for this partial scan, even if those surfaces were previously only seen in a different configuration. We evaluate the algorithm on several real-world datasets, where we demonstrate good results in the presence of significant movement of articulated parts and non-rigid surface deformation. Finally, we show that the output of the algorithm can be used for compelling computer graphics tasks such as interpolation between two scans of a non-rigid object and automatic recovery of articulated object models.

## 1 Introduction

The construction of 3D object models is a key task for many graphics applications. It is becoming increasingly common to acquire these models from a range scan of a physical object. This paper deals with an important subproblem of this acquisition task — the problem of registering two deforming surfaces corresponding to different configurations of the same non-rigid object.

The main difficulty in the 3D registration problem is determining the *correspondences* of points on one surface to points on the other. Local regions on the surface are rarely distinctive enough to determine the correct correspondence, whether because of noise in the scans, or because of symmetries in the object shape. Thus, the set of candidate correspondences to a given point is usually large. Determining the correspondence for all object points results in a combinatorially large search problem. The existing algorithms for deformable surface

---

*A results video is available at *http://robotics.stanford.edu/∼drago/cc/video.mp4*

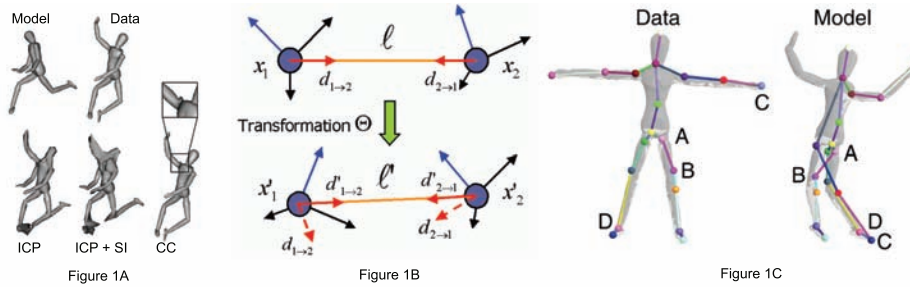

Figure 1: A) Registration results for two meshes. Nonrigid ICP and its variant augmented with spin images get stuck in local maxima. Our CC algorithm produces a largely correct registration, although with an artifact in the right shoulder (inset). B) Illustration of the link deformation process C) The CC algorithm which uses only deformation potentials can violate mesh geometry. Near regions can map to far ones (segment AB) and far regions can map to near ones (points C,D).

registration make the problem tractable by assuming significant prior knowledge about the objects being registered. Some rely on the presence of markers on the object [1, 20], while others assume prior knowledge about the object dynamics [16], or about the space of non-rigid deformations [15, 5]. Algorithms that make neither restriction [18, 12] simplify the problem by decorrelating the choice of correspondences for the different points in the scan. However, this approximation is only good in the case when the object deformation is small; otherwise, it results in poor local maxima as nearby points in one scan are allowed to map to far-away points in the other.

Our algorithm defines a joint probabilistic model over all correspondences, which explicitly model the correlations between them — specifically, that nearby points in one mesh should map to nearby points in the other. Importantly, the notion of "nearby" used in our model is defined in terms of geodesic distance over the mesh. We define a probabilistic model over the set of correspondences, that encodes these geodesic distance constraints as well as penalties for link twisting and stretching, and high-level local surface features [14]. We then apply *loopy belief propagation* [21] to this model, in order to solve for the entire set of correspondences simultaneously. The result is a registration that respects the surface geometry. To the best of our knowledge, the algorithm we present in this paper is the first algorithm which allows the registration of 3D surfaces of an object where the object configurations can vary significantly, there is no prior knowledge about object shape or dynamics of deformation, and nothing whatsoever is known about the object alignment. Moreover, unlike many methods, our algorithm can be used to register a partial scan to a complete model, greatly increasing its applicability.

We apply our approach to three datasets containing 3D scans of a wooden puppet, a human arm and entire human bodies in different configurations. We demonstrate good registration results for scan pairs exhibiting articulated motion, non-rigid deformations, or both. We also describe three applications of our method. In our first application, we show how a partial scan of an object can be registered onto a fully specified model in a different configuration. The resulting registration allows us to use the model to "complete" the partial scan in a way that preserves the local surface geometry. In the second, we use the correspondences found by our algorithm to smoothly interpolate between two different poses of an object. In our final application, we use a set of registered scans of the same object in different positions to recover a decomposition of the object into approximately rigid parts, and recover an articulated skeleton linking the parts. All of these applications are done in an unsupervised way, using only the output of our Correlated Correspondence algorithm applied to pairs of poses with widely varying deformations, and unknown initial alignments. These results demonstrate the value of a high-quality solution to the registration problem to a range of graphics tasks.

## 2  Previous Work

Surface registration is a fundamental building block in computer graphics. The classical solution for registering rigid surfaces is the Iterative Closest Point algorithm (ICP) [4, 6, 17]. Recently, there has been work extending ICP to non-rigid surfaces [18, 8, 12, 1]. These algorithms treat one of the scans (usually a complete model of the surface) as a deformable template. The links between adjacent points on the surface can be thought of as springs, which are allowed to deform at a cost. Similarly to ICP, these algorithms iterate between two subproblems — estimating the non-rigid transformation $\Theta$ and estimating the set of correspondences $C$ between the scans. The step estimating the correspondences assumes that a good estimate of the nonrigid transformation $\Theta$ is available. Under this assumption, the assignments to the correspondence variables become decorrelated: each point in the second scan is associated with the nearest point (in the Euclidean distance sense) in the deformed template scan. However, the decomposition also induces the algorithm's main limitation. By assigning points in the second scan to points on the deformed model independently, nearby points in the scan can get associated to remote points in the model if the estimate of $\Theta$ is poor (Fig. 1A). While several approaches have been proposed to address this problem of incorrect correspondences, their applicability is largely limited to problems where the deformation is local, and the initial alignment is approximately correct.

Another line of related work is the work on deformable template matching in the computer vision community. In the 3D case, this framework is used for detection of articulated object models in images [13, 22, 19]. The algorithms assume the decomposition of the object into a relatively small number of parts is known, and that a detector for each object part is available. Template matching approaches have also been applied to deformable 2D objects, where very efficient solutions exist [9, 11]. However, these methods do not extend easily to the case of 3D surfaces.

## 3  The Correlated Correspondence Algorithm

The input to the algorithm is a set of two meshes (surfaces tessellated into polygons). The *model mesh* $X = (V^X, E^X)$ is a complete model of the object, in a particular pose. $V^X = (x_1, \ldots, x_N)$ denotes the mesh points, while $E^X$ is the set of *links* between adjacent points on the mesh surface. The *data mesh* $Z = (V^Z, E^Z)$ is either a complete model or a partial view of the object in a different configuration. Each data mesh point $z_k$ is associated with a *correspondence variable* $c_k$, specifying the corresponding model mesh point. The task of registration is one of estimating the set of all correspondences $C$ and a non-rigid transformation $\Theta$ which aligns the corresponding points.

### 3.1  Probabilistic Model

We formulate the registration problem as one of finding an embedding of the data mesh $Z$ into the model mesh $X$, which is encoded as an assignment to all correspondence variables $C = (c_1, \ldots, c_K)$. The main idea behind our approach is to preserve the consistency of the embedding by explicitly correlating the assignments to the correspondence variables. We define a joint distribution over the correspondence variables $c_1, \ldots, c_K$, represented as a Markov network. For each pair of adjacent data mesh points $z_k, z_l$, we want to define a probabilistic potential $\psi(c_k, c_l)$ that constrains this pair of correspondences to reasonable and consistent. This gives rise to a joint probability distribution of the form $p(C) = \frac{1}{Z} \prod_k \psi(c_k) \prod_{k,l} \psi(c_k, c_l)$ which contains only single and pairwise potentials. Performing probabilistic inference to find the most likely joint assignment to the entire set of correspondence variables $C$ should yield a good and consistent registration.

**Deformation Potentials.**  We want our model to encode a preference for embeddings of mesh $Z$ into mesh $X$, which minimize the amount of deformation $\Theta$ induced by the embedding. In order to quantify the amount of deformation $\Theta$, applied to the model, we

will follow the ideas of Hähnel *et al.* [12] and treat the links in the set $E^X$ as springs, which resist stretching and twisting at their endpoints. Stretching is easily quantified by looking at changes in the link length induced by the transformation $\Theta$. Link twisting, however, is ill-specified by looking only at the Cartesian coordinates of the points alone. Following [12], we attach an imaginary *local coordinate system* to each point on the model. This local coordinate system allows us to quantify the "twist" of a point $x_j$ relative to a neighbor $x_i$. A non-rigid transformation $\Theta$ defines, for each point $x_i$, a translation of its coordinates and a rotation of its local coordinate system.

To evaluate the deformation penalty, we parameterize each link in the model in terms of its length and its direction relative to its endpoints (see Fig. 1B). Specifically, we define $l_{i,j}$ to be the distance between $x_i$ and $x_j$; $d_{i \to j}$ is a unit vector denoting the direction of the point $x_j$ in the coordinate system of $x_i$ (and vice versa). We use $e_{i,j}$ to denote the set of edge parameters $(l_{i,j}, d_{i \to j}, d_{j \to i})$. It is now straightforward to specify the penalty for model deformations. Let $\Theta$ be a transformation, and let $\tilde{e}_{i,j}$ denote the triple of parameters associated with the link between $x_i$ and $x_j$ after applying $\Theta$. Our model penalizes twisting and stretching, using a separate zero-mean Gaussian noise model for each:

$$P(\tilde{e}_{i,j} \mid e_{i,j}) = P(\tilde{l}_{i,j} \mid l_{i,j}) \, P(\tilde{d}_{i \to j} \mid d_{i \to j}) \, P(\tilde{d}_{j \to i} \mid d_{j \to i}) \qquad (1)$$

In the absence of prior information, we assume that all links are equally likely to deform.

In order to quantify the deformation induced by an embedding $C$, we need to include a potential $\psi_d(c_k, c_l)$ for each link $e_{k,l}^Z \in E^Z$. Every probability $\psi_d(c_k = i, c_l = j)$ corresponds to the deformation penalty incurred by deforming model link $e_{i,j}$ to generate link $e_{k,l}^Z$ and is defined in (1). We do not restrict ourselves to the set of links in $E^X$, since the original mesh tessellation is sparse and local. Any two points in $X$ are allowed to implicitly define a link.

Unfortunately, we cannot directly estimate the quantity $P(e_{k,l}^Z \mid e_{i,j})$, since the link parameters $e_{k,l}^Z$ depend on knowing the nonrigid transformation, which is not given as part of the input. The key issue is estimating the (unknown) relative rotation of the link endpoints. In effect, this rotation is an additional latent variable, which must also be part of the probabilistic model. To remain within the realm of discrete Markov networks, allowing the application of standard probabilistic inference algorithms, we discretize the space of the possible rotations, and fold it into the domains of the correspondence variables. For each possible value of the correspondence variable $c_k = i$ we select a small set of candidate rotations, consistent with local geometry. We do this by aligning local patches around the points $x_i$ and $z_k$ using rigid ICP. We extend the domain of each correspondence variables $c_k$, where each value encodes a matching point *and* a particular rotation from the precomputed set for that point. Now the edge parameters $e_{k,l}^Z$ are fully determined and so is the probabilistic potential.

**Geodesic Distances.** Our proposed approach raises the question as to what constitutes the best constraint between neighboring correspondence variables. The literature on scan registration — for rigid and non-rigid models alike — relies on the preserving Euclidean distance. While Euclidean distance is meaningful for rigid objects, it is very sensitive to deformations, especially those induced by moving parts. For example, in Fig. 1C, we see that the two legs in one configuration of our puppet are fairly close together, allowing the algorithm to map two adjacent points in the data mesh to the two separate legs, with minimal deformation penalty. In the complementary situation, especially when object symmetries are present, two distant yet similar points in one scan might get mapped to the same region in the other. For example, in the same figure, we see that points in both an arm and a leg in the data mesh get mapped to a single leg in the model mesh.

We therefore want to enforce constraints preserving distance along the mesh surface (geodesic distance). Our probabilistic framework easily incorporate such constraints as correlations between pairs of correspondence variables. We encode a *nearness preservation*

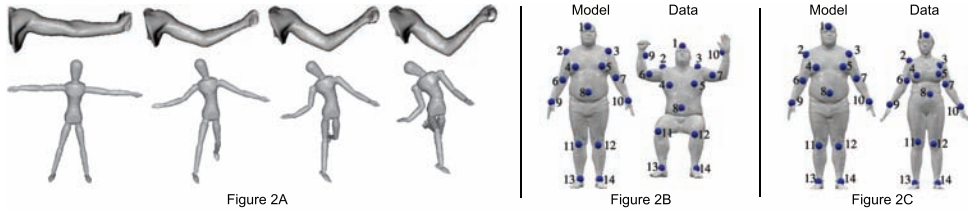

Figure 2: A) Automatic interpolation between two scans of an arm and a wooden puppet. B) Registration results on two scans of the same man sitting and standing up (select points were displayed) C) Registration results on scans of a larger man and a smaller woman. The algorithm is robust to small changes in object scale.

*constraint* which prevents adjacent points in mesh $Z$ to be mapped to distant points in $X$ in the geodesic distance sense. For *adjacent* points $z_k, z_l$ in the data mesh, we define the following potential:

$$\psi_n(c_k = i, c_l = j) = \begin{cases} 0 & \text{dist}_{\text{Geodesic}}(x_i, x_j) > \alpha\rho \\ 1 & \text{otherwise} \end{cases} \qquad (2)$$

where $\rho$ is the data mesh resolution and $\alpha$ is some constant, chosen to be 3.5.

The *farness preservation* potentials encode the complementary constraint. For *every* pair of points $z_k, z_l$ whose geodesic distance is more than $5\rho$ on the data mesh, we have a potential:

$$\psi_f(c_k = i, c_l = j) = \begin{cases} 0 & \text{dist}_{\text{Geodesic}}(x_i, x_j) < \beta\rho \\ 1 & \text{otherwise} \end{cases} \qquad (3)$$

where $\beta$ is also a constant, chosen to be 2 in our implementation. The intuition behind this constraint is fairly clear: if $z_k, z_l$ are far apart on the data mesh, then their corresponding points must be far apart on the model mesh.

**Local Surface Signatures.** Finally, we encode a set of potentials that correspond to the preservation of local surface properties between the model mesh and data mesh. The use of local surface signatures is important, because it helps to guide the optimization in the exponential space of assignments. We use spin images [14] compressed with principal component analysis to produce a low-dimensional *signature* $s_x$ of the local surface geometry around a point $x$. When data and model points correspond, we expect their local signatures to be similar. We introduce a potential whose values $\psi_s(c_k) = i$ enforce a zero-mean Gaussian penalty for discrepancies between $s_{x_i}$ and $s_{z_k}$.

## 3.2 Optimization

In the previous section, we defined a Markov network, which encodes a joint probability distribution over the correspondence variables as a product of single and pairwise potentials. Our goal is to find a joint assignment to these variables that maximizes this probability. This problem is one of standard probabilistic inference over the Markov network. However, the Markov network is quite large, and contains a large number of loops, so that exact inference is computationally infeasible. We therefore apply an approximate inference method known as *loopy belief propagation (LBP)*[21], which has been shown to work in a wide variety of applications. Running LBP until convergence results in a set of probabilistic assignments to the different correspondence variables, which are locally consistent. We then simply extract the most likely assignment for each variable to obtain a correspondence.

One remaining complication arises from the form of our farness preservation constraints. In general, most pairs of points in the mesh are not close, so that the total number of such potentials grows as $O(M^2)$, where $M$ is the number of points in the data mesh. However, rather than introducing all these potentials into the Markov net from the start, we

introduce them as needed. First, we run LBP without any farness preservation potentials. If the solution violates a set of farness preservation constraints, we add it and rerun BP. In practice, this approach adds a very small number of such constraints.

## 4    Experimental Results

**Basic Registration.**    We applied our registration algorithm to three different datasets, containing meshes of a human arm, wooden puppet and the CAESAR dataset of whole human bodies [1], all acquired by a 3D range scanner. The meshes were not complete surfaces, but several techniques exist for filling the holes (e.g., [10]).

We ran the Correlated Correspondence algorithm using the same probabilistic model and the same parameters on all data sets. We use a coarse-to-fine strategy, using the result of a coarse sub-sampling of the mesh surface to constrain the correspondences at a finer-grained level. The resulting set of correspondences were used as markers to initialize the non-rigid ICP algorithm of Hähnel *et al.* [12].

The Correlated Correspondence algorithm successfully aligned all mesh pairs in our human arm data set containing 7 arms. In the puppet data set we registered one of the meshes to the remaining 6 puppets. The algorithm correctly registered 4 out of 6 data meshes to the model mesh. In the two remaining cases, the algorithm produced a registration where the torso was flipped, so that the front was mapped to the back. This problem arises from ambiguities induced by the puppet symmetry, whose front and back are almost identical. Importantly, our probabilistic model assigns a higher likelihood score to the correct solution, so that the incorrect registration is a consequence of local maxima in the LBP algorithm.

This fact allows us to address the issue in an unsupervised way simply by running loopy BP several times, with different initialization. For details on the unsupervised initialization scheme we used, please refer to our technical report [2]. We ran the modified algorithm to register one puppet mesh to the remaining 6 meshes in the dataset, obtaining the correct registration in all cases. In particular, as shown in Fig. 1A, we successfully deal with the case on which the straightforward nonrigid ICP algorithm failed. The modified algorithm was applied to the CAESAR dataset and produced very good registration for challenging cases exhibiting both articulated motion and deformation (Fig. 2B), or exhibiting deformation and a (small) change in object scale (Fig. 2C).

Overall, the algorithm performed robustly, producing a close-to-optimal registrations even for pairs of meshes that involve large deformations, articulated motion or both. The registration is accomplished in an unsupervised way, without any prior knowledge about object shape, dynamics, or alignment.

**Partial view completion.**    The Correlated Correspondence algorithm allows us to register a data mesh containing only a partial scan of an object to a known complete surface model of the object, which serves as a template. We can then transform the template mesh to the partial scan, a process which leaves undisturbed the links that are not involved in the partial mesh. The result is a mesh that matches the data on the observed points, while completing the unknown portion of the surface using the template.

We take a partial mesh, which is missing the entire back part of the puppet in a particular pose. The resulting partial model is displayed in Fig. 3B-1; for comparison, the correct complete model in this configuration (which was not available to the algorithm), is shown in Fig. 3B-2. We register the partial mesh to models of the object in a different pose (Fig. 3B-3), and compare the completions we obtain (Fig. 3B-4), to the ground truth represented in Fig. 3B-2. The result demonstrates a largely correct reconstruction of the complete surface geometry from the partial scan and the deformed template. We report additional shape completion results in [2].

**Interpolation.**    Current research [20] shows that if a nonrigid transformation $\Theta$ between the poses is available, believable animation can be produced by linear interpolation be-

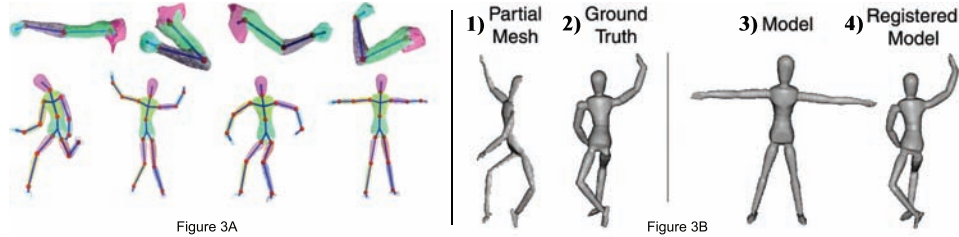

Figure 3: A) The results produced by the CC algorithm were used for unsupervised recovery of articulated models. 15 puppet parts and 4 arm parts, as well as the articulated object skeletons, were recovered. B) Partial view completion results. The missing parts of the surface were estimated by registering the partial view to a complete model of the object in a different configuration.

tween the model mesh and the transformed model mesh. The interpolation is performed in the space of local link parameters $(l_{i,j}, d_{i \rightarrow j}, d_{j \rightarrow i})$, We demonstrate that transformation estimates produced by our algorithm can be used to automatically generate believable animation sequences between fairly different poses, as shown in Fig. 2A.

**Recovering Articulated Models.** Articulated object models have a number of applications in animation and motion capture, and there has been work on recovering them automatically from 3D data [7, 3]. We show that our unsupervised registration capability can greatly assist articulated model recovery. In particular, the algorithm in [3] requires an estimate of the correspondences between a template mesh and the remaining meshes in the dataset. We supplied it with registration computed with the Correlated Correspondence algorithm. As a result we managed to recover in a completely unsupervised way all 15 rigid parts of the puppet, as well as the joints between them (Fig. 3A). We demonstrate successful articulation recovery even for objects which are not purely rigid, as is the case with the human arm (see Fig. 3A).

## 5   Conclusion

The contribution of this paper is an algorithm for unsupervised registration of non-rigid 3D surfaces in significantly different configurations. Our results show that the algorithm can deal with articulated objects subject to large joint movements, as well as with non-rigid surface deformations. The algorithm was not provided with markers or other cues regarding correspondence, and makes no assumptions about object shape, dynamics, or alignment. We show the quality and the utility of the registration results we obtain by using them as a starting point for compelling computer graphics applications: partial view completion, interpolation between scans, and recovery of articulated object models. Importantly, all these results were generated in a completely unsupervised manner from a set of input meshes.

The main limitation of our approach is the fact that it makes the assumption of (approximate) preservation of geodesic distance. Although this assumption is desirable in many cases, it is not always warranted. In some cases, the mesh topology may change drastically, for example, when an arm touches the body. We can try to extend our approach to handle these cases by trying to detect when they arise, and eliminating the associated constraints. However, even this solution is likely to fail on some cases. A second limitation of our approach is that it assumes that the data mesh is a subset of the model mesh. If the data mesh contains clutter, our algorithm will attempt to embed the clutter into the model. We feel that the general nonrigid registration problem becomes underspecified when significant clutter and occlusion are present simultaneously. In this case, additional assumptions about the surfaces will be needed.

Despite the fact that our algorithm performs quite well, there are limitations to what can be accurately inferred about the object from just two scans. Given more scans of the

same object, we can try to learn the deformation penalty associated with different links, and bootstrap the algorithm. Such an extension would be a step toward the goal of learning models of object shape and dynamics from raw data.

**Acknowledgments.** This work has been supported by the ONR Young Investigator (PECASE) grant N00014-99-1-0464, and ONR Grant N00014-00-1-0637 under the DoD MURI program.

# References

[1] B Allen, B Curless, and Z Popovic. The space of human body shapes:reconstruction and parameterization from range scans. In *Proc. SIGGRAPH*, 2003.

[2] D. Anguelov, D.Koller, P. Srinivasan, S.Thrun, H. Pang, and J.Davis. The correlated correspondence algorithm for unsupervised registration of nonrigid surfaces. In *TR-SAIL-2004-100, at* `http://robotics.stanford.edu/~drago/cc/tr100.pdf`, 2004.

[3] D. Anguelov, D. Koller, H. Pang, P. Srinivasan, and S. Thrun. Recovering articulated object models from 3d range data. In *Proc. UAI*, 2004.

[4] P. Besl and N. McKay. A method for registration of 3d shapes. *Transactions on Pattern Analysis and Machine Intelligence*, 14(2):239–256, 1992.

[5] V Blanz and T Vetter. A morphable model for the synthesis of 3d faces. In *SIGGRAPH*, 1999.

[6] Y. Chen and G. Medioni. Object modeling by registration of multiple range images. In *Proc. IEEE Conf. on Robotics and Automation*, 1991.

[7] K. Cheung, S. Baker, and T. Kanade. Shape-from-silhouette of articulated objects and its use for human body kinematics estimation and motion capture. In *Proc. IEEE CVPR*, 2003.

[8] H. Chui and A. Rangarajan. A new point matching algorithm for non-rigid registration. In *Proceedings of the Conference on Computer Vision and Pattern Recognition (CVPR)*, 2000.

[9] J. Coughlan and S. Ferreira. Finding deformable shapes using loopy belief propagation. In *Proc. ECCV*, volume 3, pages 453–468, 2002.

[10] J. Davis, S. Marschner, M. Garr, and M. Levoy. Filling holes in complex surfaces using volumetric diffusion. In *Symposium on 3D Data Processing, Visualization, and Transmission*, 2002.

[11] Pedro Felzenszwalb. Representation and detection of shapes in images. In *PhD Thesis*. Massachusetts Institute of Technology, 2003.

[12] D. Hähnel, S. Thrun, and W. Burgard. An extension of the ICP algorithm for modeling nonrigid objects with mobile robots. In *Proc. IJCAI*, Acapulco, Mexico, 2003.

[13] D. Huttenlocher and P. Felzenszwalb. Efficient matching of pictorial structures. In *CVPR*, 2003.

[14] Andrew Johnson. *Spin-Images: A Representation for 3-D Surface Matching*. PhD thesis, Robotics Institute, Carnegie Mellon University, Pittsburgh, PA, August 1997.

[15] Michael Leventon. Statistic models in medical image analysis. In *PhD Thesis*. Massachusetts Institute of Technology, 2000.

[16] Michael H. Lin. Tracking articulated objects in real-time range image sequences. In *ICCV (1)*, pages 648–653, 1999.

[17] S. Rusinkiewicz and M. Levoy. Efficient variants of the ICP algorithm. In *Proc. 3DIM*, Quebec City, Canada, 2001. IEEEComputer Society.

[18] Christian Shelton. Morphable surface models. In *International Journal of Computer Vision*, 2000.

[19] Leonid Sigal, Michael Isard, Benjamin H. Sigelman, and Michael J. Black. Attractive people: Assembling loose-limbed models using non-parametric belief propagation. In *NIPS*, 2003.

[20] R. Sumner and Jovan Popović. Deformation transfer for triangle meshes. In *SIGGRAPH*, 2004.

[21] J. Yedidia, W. Freeman, and Y Weiss. Understanding belief propagation and its generalizations. In *Exploring Artificial Intelligence in the New Millennium*. Science & Technology Books, 2003.

[22] S. Yu, R. Gross, and J. Shi. Concurrent object recognition and segmentation with graph partitioning. In *Proc. NIPS*, 2002.
